# Norepinephrine and Neural Interrupts

**Peter Dayan**
Gatsby Computational Neuroscience Unit
University College London
17 Queen Square, London WC1N 3AR, UK
dayan@gatsby.ucl.ac.uk

**Angela J. Yu**
Center for Brain, Mind & Behavior
Green Hall, Princeton University
Princeton, NJ 08540, USA
ajyu@princeton.edu

## Abstract

Experimental data indicate that norepinephrine is critically involved in aspects of vigilance and attention. Previously, we considered the function of this neuromodulatory system on a time scale of minutes and longer, and suggested that it signals global uncertainty arising from gross changes in environmental contingencies. However, norepinephrine is also known to be activated phasically by familiar stimuli in well-learned tasks. Here, we extend our uncertainty-based treatment of norepinephrine to this phasic mode, proposing that it is involved in the detection and reaction to state uncertainty *within* a task. This role of norepinephrine can be understood through the metaphor of neural interrupts.

## 1 Introduction

Theoretical approaches to understanding neuromodulatory systems are plagued by the latter's neural ubiquity, evolutionary longevity, and temporal promiscuity. Neuromodulators act in potentially different ways over many different time-scales [14]. There are various general notions about their roles, such as regulating sleeping and waking [13] and changing the signal to noise ratios of cortical neurons [11]. However, these are slowly giving way to more specific computational ideas [20, 7, 10, 24, 25, 5], based on such notions as optimal gain scheduling, prediction error and uncertainty.

In this paper, we focus on the short term activity of norepinephrine (NE) neurons in the locus coeruleus [18, 1, 2, 3, 16, 4]. These neurons project NE to subcortical structures and throughout the entire cortex, with individual neurons having massive axonal arborizations [12]. Over medium and short time-scales, norepinephrine is implicated in various ways in attention, vigilance, and learning. Given the widespread distribution and effects of NE in key cognitive tasks, it is very important to understand what it is in a task that drives the activity of NE neurons, and thus what computational effects it may be exerting.

Figure 1 illustrates some of the key data that has motivated theoretical treatments of NE. Figure 1A;B;C show more tonic responses operating around a time-scale of minutes. Figures 1D;E;F show the short-term effects that are our main focus here.

Briefly, Figures 1A;B show that when the rules of a task are reversed, NE influences the speed of adaptation to the changed contingency (Figure 1A) and the activity of noradrenergic cells is tonically elevated (Figure 1B). Based on these data, we suggested [24, 25] that medium-term NE reports *unexpected uncertainty* arising from unpredicted changes in an environment or task. This signal is a key part of a strategy for inference in potentially labile contexts. It operates in collaboration with a putatively cholinergic signal which reports on *expected uncertainty* that arises, for instance, from known variability or noise.

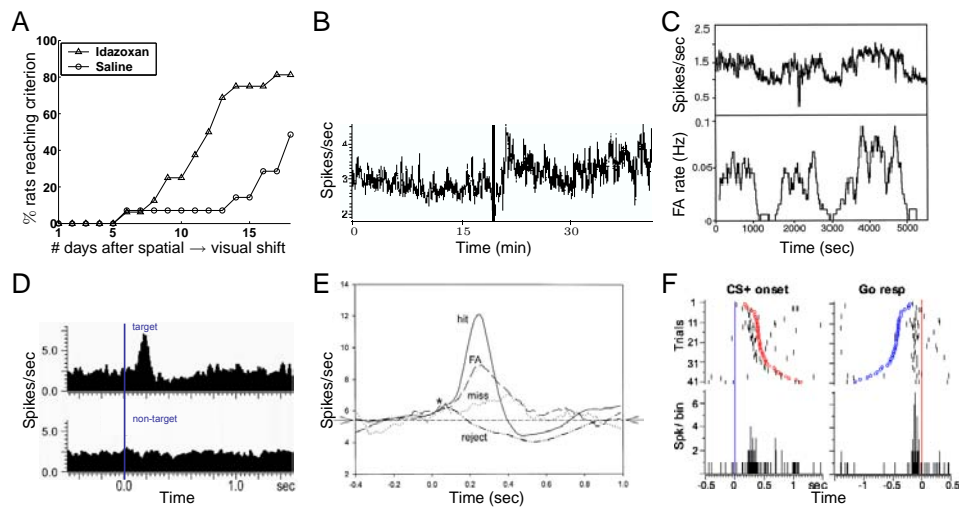

Figure 1: NE activity and effects. (A) Rats solve a sequential decision problem in a linear maze. When the relevant cues are switched after a few days of learning (from spatial to visual), rats with pharmacologically boosted NE ("idazoxan") learn to use the new set of cues *faster* than the controls. Adapted from [9]. (B) In a vigilance task, monkeys respond to rare targets and ignore common distractor stimuli. The trace shows the activity of a single NE neuron in the locus coeruleus (LC) around the time of a target-distractor reversal (vertical line). Tonic activity is elevated for a considerable period. Adapted from [2]. (C) Correlation between the gross fluctuations in the tonic activity of a single NE neuron (upper) and performance in the task (lower, measured by false alarm rate). Adapted from [20]. (D) Single NE cells are activated on a phasic time-scale stimulus locked (vertical line) to the target (upper plot) and not the distractor (lower plot). Adapted from [16]. (E) The average responses of a large number of norepinephrine cells (over a total of 41,454 trials) stimulus locked (vertical line) to targets or distractors, sorted by the nature and rectitude of the response. The asterisk marks (similar) early activation of the neurons by the stimulus. Adapted from [16]. (F) In a GO/NO-GO olfactory discrimination task for rats, single units are activated by the target odor (and not by the distractor odor), but are temporally much more tightly locked to the response (right) than the stimulus (left). Trials are ordered according to the time between stimulus (blue) and response (red). Adapted from [4].

However, Figures 1D;E;F, along with other substantial neurophysiological data on the activity of NE neurons [18, 4], show NE neurons have phasic response properties that lie outside this model. The data in Figure 1D;E come from a vigilance task [1], in which subjects can gain reward by reacting to a rare target (a rectangle oriented one way), while ignoring distractors (a rectangle oriented in the orthogonal direction). Under these circumstances, NE is consistently activated by the target and *not* the distractor (Figure 1D). There are also clear correlations in the magnitude of the NE activity and the nature of a trial: hit, miss, false alarm, correct reject (Figure 1E). It is known that the activity is weaker if the targets are more common [17] (though the lack of response to rare distractors shows that NE is not driven by mere rarity), and disappears if no action need be taken in response to the target [18]. In fact, the signal is more tightly related in time to the subsequent action than the preceding stimulus (Figure 1F). The signal has been qualitatively described in terms of influencing or controlling the allocation of behavioral or cognitive resources [20, 4].

Since it arises on every trial in an extremely well-learned task with stable stimulus contingencies, this NE signal clearly cannot be indicating unpredicted task changes. Brown *et*

*al* [5] have recently made the seminal suggestion that it reports changes in the statistical structure of the input (stimulus-present versus stimulus-absent) to decision-making circuits that are involved in initiating differential responding to distinct target stimuli. A statistically necessary consequence of the change in the input structure is that afferent information should be integrated differently: sensory responses should be ignored if no target is present, but taken seriously otherwise. Their suggestion is that NE, by changing the gain of neurons in the decision-making circuit, has exactly this optimizing effect.

In this paper, we argue for a related, but distinct, notion of phasic NE, suggesting that it reports on unexpected *state* changes within a task. This is a significant, though natural, extension of its role in reporting unexpected *task* changes [25]. We demonstrate that it accounts well for the neurophysiological data. In agreement with the various accounts of the effects of phasic NE, we consider its role as a form of internal *interrupt* signal [6]. Computers use interrupts to organize the correct handling of internal and external events such as timers or peripheral input. Higher-level programs specify what interrupts are allowed to gain control, and the consequences thereof. We argue that phasic NE is the medium for a somewhat similar neural interrupt, allowing the correct handling of statistically *atypical* events. This notion relates comfortably to many existing views of phasic NE, and provides a computational correlate for quantitative models.

## 2  The Model

Figure 2A illustrates a simple hidden Markov generative model (HMM) of the vigilance task in Figure 1B-E. The (start) state models the condition established when the monkey fixates the light and initiates a trial. Following a somewhat variable delay, either the target (target) or the distractor (distractor) is presented, and the monkey must respond appropriately (release a continuously depressed bar for target and continue pressing for distractor) The transition out of start is uniformly distributed between timesteps 6 and 10, implemented by a time-varying transition function for this node:

$$P(s_t|s_{t-1} = \text{start}) = \begin{cases} 1 - q_t & s_t = \text{start} \\ 0.8q_t & s_t = \text{distractor} \\ 0.2q_t & s_t = \text{target} \end{cases} \qquad (1)$$

where $q_t = 1/(11-t)$ for ($6 \le t \le 10$) and $q_t = 0$ otherwise. The start and target states are assumed to be absorbing states (self-transition probability = 1). This transition function ensures that the stimulus onset has a uniform distribution between 6 and 10 timesteps (and 0 otherwise). Given that a transition out of start (into either target or distractor) takes place, the probability is .2 for entering target and .8 for start, as in the actual task.

In addition, it is assumed that the node start does not emit observations, while target emits $x_t = $ t with probability $\eta > 0.5$ and d with probability $1 - \eta$, and distractor emits $x_t = $ d with probability $\eta$ and t with probability $1 - \eta$. The transition out of start is evident as soon as the first d or t is observed, while the magnitude of $\eta$ controls the "confusability" of the target and distractor states. Figure 2B shows a typical run from this generative model. The transition into target happens on step 10 (top), and the outputs generated are a mixture of t and d(middle), with an overall prevalence of t (bottom).

Exact inference on this model can be performed in a manner similar to the forward pass in a standard HMM:

$$P(s_t|x_1, \ldots, x_t) \propto p(x_t|s_t) \sum_{s_{t-1}} P(s_t|s_{t-1})P(s_{t-1}|x_1, \ldots, x_{t-1}) . \qquad (2)$$

Because start does not produce outputs, as soon as the first t is observed, the probability of start plummets to 0. There then ensues an inferential battle between target and distractor, with the latter having the initial advantage, since its prior probability is 80%.

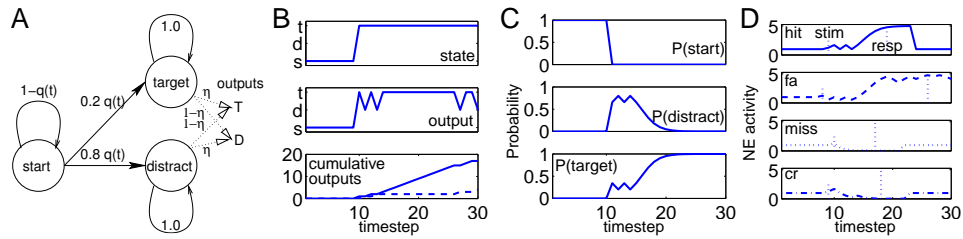

Figure 2: The model. (A) Task is modeled as a hidden Markov model (HMM), with transitions from start to either distractor (probability .8) or target (probability .2). The transitions happen between timesteps 6 and 10 with uniform probability; distractor and target are absorbing states. The only outputs are from the absorbing states, and the two have overlapping output distributions over t and d with probabilities $\eta > .5$ for their "own" output (t for target, and d for distractor), and $1-\eta$ for the other output. (B) Sample run with a transition from start to target at timestep 10 (upper). The outputs favor target once the state has changed (middle), more clearly shown in the cumulative plot (bottom). (C) Correct probabilistic inference in the task leads to the probabilities for the three states as shown. The distractor's initial advantage arises from a base rate effect, as it is the more likely default transition. (D) Model NE signal for four trials including one for hit (top; same trials as in B;C), a false alarm (fa), a miss (miss) and a correct rejection (cr). The second vertical line represents the point at which the decision was taken (target vs. distractor).

Because of the preponderance of transitions to distractor over target, the distractor state can be thought of as the *reference* or *default* state. Evidence against that default state is a form of unexpected uncertainty within a task, and we propose that phasic NE reports this uncertainty. More specifically, NE signals $P(\text{target}|x_1, \ldots, x_t)/P(\text{target})$, where $P(\text{target}) = .2$ is the prior probability of observing a target trial. We assume that a target-response is initiated when $P(s_t|x_1, \ldots, x_t)$ exceeds $0.95$, or equivalently, when the NE signal exceeds $0.95/P(\text{target})$. This implies the following intuitive relationship: the smaller the probability of the non-default state target the greater the NE-mediated "surprise" signal has to be in order to convince the inferential system that an anomalous stimulus has been observed. We also assume that if the posterior probability of target reaches $0.01$, then the trial ends with no action (either a cr or a miss). The asymmetry in the thresholds arises from the asymmetry in the response contingencies of the task. Further, to model non-inferential errors, we assume that there is probability of $0.0005$ per timestep of releasing the bar after the transition out of start. Once a decision is reached, the NE signal is set back to baseline (1, for equal prior and posterior) after a delay of $5$ timesteps.

Note that the precise form of the mapping from unexpected uncertainty to NE spikes is rather arbitrary. In particular, there may be a strong non-linearity, such as a thresholded response profile. For simplicity, we assume a linear mapping between the two.

The NE activity during the start state is also rather arbitrary. Activity is at baseline before the stimulus comes on, since prior and posterior match when there is no explicit information from the world. When the stimulus comes on, the divisive normalization makes the activity go above baseline because although the transition was expected, its occurrence was not predicted with perfect precision. The magnitude of this activity depends on the precision of the model of the time of the transition; and the uncertainty in the interval timer. We set it to a small super-baseline level to match the data.

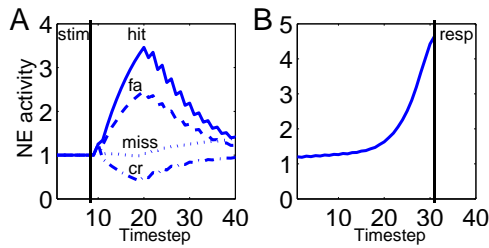

Figure 3: NE activity. (A) NE activity locked to the stimulus onset (*ie* the transition out of start). (B) NE activity response-locked to the decision to act, just for hit and fa trials. Note the difference in scale between the two figures.

## 3  Results

Figure 2C illustrates the inferential performance of the model for the sample run in Figure 2B;C. When the first t is observed on timestep 10, the probability of start drops to 0 and the probability of distractor, which has an initial advantage over target due to its higher probability, eventually loses out to target as the evidence overwhelms the prior. Figure 2D shows the model's NE signal for one example each of hit, fa, miss, and cr trials.

Figure 3 presents our main results. Figure 3A shows the average NE signal for the four classes of responses (hit, false alarm, miss, and correct rejection), time-locked to the start of the stimulus. These traces should be compared with those in Figure 1E. The basic form of the *rise* of the signal in the model is broadly similar to that in the data; as we have argued, the fall is rather arbitrary. Figure 3B shows the average signal locked to the time of reaction (for hit and false alarm trials) rather than stimulus onset. As in the data (Figure 1F), response-locked activities are much more tightly clustered, although this flatters the model somewhat, since we do not allow for any variability in the response time as a function of when the probability of state target reaches the threshold. Since the decay of the signal following a response is unconstrained, the trace terminates when the response is determined, usually when the probability of target reaches threshold, but also sometimes when there is an accidental erroneous response.

Figure 4 shows some additional features of the NE signal in this case. Figure 4A compares the effect of making the discrimination between target and distractor more or less difficult in the model (upper) and in the data (lower; [16]). As in the data, the stimulus-locked NE signal is somewhat broader for the more difficult case, since information has to build up over a longer period. Also as in the data, correct rejections are much less affected than hits. Figure 4B shows response locked NE. Although it is correctly slightly broader for the more difficult discrimination, the timing is not quite the same. This is largely due to the lack of a realistic model tying the defeat of the default state assumption to a behavioral response. For the easy task ($\eta = 0.675$), there were $19\%$ hits, $1.5\%$ false alarms, $1\%$ misses and $77\%$ correct rejections. For the difficult task ($\eta = 0.65$) the main difference was an increase in the number of misses to $1.5\%$, largely at the expense of hits. Note that since the NE signal is calculated relative to the prior likelihood, making target *more* likely would *reduce* the NE signal exactly proportionally. The data certainly hint at such a reduction [17] although the precise proportionality is not clear.

## 4  Discussion

The present model of the phasic activity of NE cells is a direct and major extension of our previous model of tonic aspects of this neuromodulator. The key difference is that

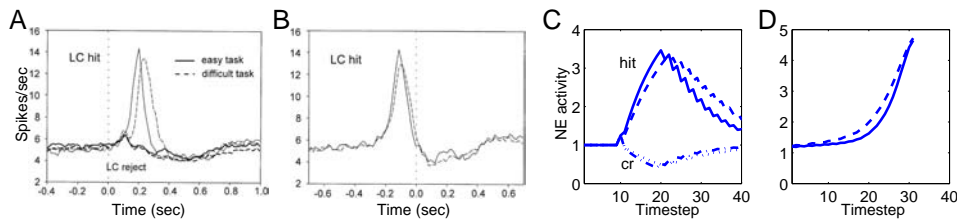

Figure 4: NE activities and task difficulty. (A) Stimulus-locked LC responses are slower and broader for a more difficult discrimination; where difficulty is controlled by the similarity of target and distractor stimuli. (B) When aligned to response, LC activities for easy and difficult discriminations are more similar, although their response in the more difficult condition is still somewhat attenuated compared to the easy one. Data in A;B adapted from [16]. (C) Discrimination difficulty in the model is controlled by the parameter $\eta$. When $\eta$ is reduced from $0.675$ (easy; solid) to $0.65$ (hard; dashed), simulated NE activity also becomes slower and broader when aligned to stimulus. (D) Same traces aligned to response indicate NE activity in the difficult condition is attenuated in the model.

unexpected uncertainty is now about the state *within* a current characterization of the task rather than about the characterization as a whole. These aspects of NE functionality are likely quite widespread, and allow us to account for a much wider range of data on this neuromodulator.

In the model, NE activity is explicitly normalized by prior probabilities arising from the default state transitions in tasks. This is necessary to measure specifically *unexpected* uncertainty, and explains the decrement in NE phasic response as a function of the target probability [17]. It is also associated with the small activation to the stimulus onset, although the precise form of this deserves closer scrutiny. For instance, if subjects were to build a richer model of the statistics of the time of the transition out of the start state, then we should see this reflected directly in the NE signal even before the stimulus comes on, for instance related to the inverse of the survival function for the transition. We would also expect this transition to effect a different NE signature if stimuli were expected during start that could also be confused with those expected during target and distractor.

If NE indeed reports on the failure of the current state within the model of the task to account successfully for the observations, then what effect should it have? One useful way to think about the signal is in terms of an *interrupt* signal in computers. In these, a control program establishes a set of conditions (*eg* keyboard input) under which normal processing should be interrupted, in order that the consequence of the interrupt (*eg* a keystroke) can be appropriately handled. Computers have highly centralized processing architecture, and therefore the interrupt signal only needs a very limited spatial extent to exert a widespread effect on the course of computation. By contrast, processing in the brain is highly distributed, and therefore it is necessary for the interrupt signal to have a widespread distribution, so that the full ramifications of the failure of the current state can be felt. Neuromodulatory systems are ideal vehicles for the signal.

The interrupt signal should engage mechanisms for establishing the new state, which then allows a new set of conditions to be established as to which interrupts will be allowed to occur, and also to take any appropriate action (as in the task we modeled). The interrupt signal can be expected to be beneficial, for instance, when there is competition between tasks for the use of neural resources such as receptive fields [8].

Apart from interrupts such as these under sophisticated top-down control, there are also more basic contingencies from things such as critical potential threats and stressors that

should exert a rapid and dramatic effect on neural processing (these also have computational analogues in signals such as that power is about to fail). The NE system is duly subject to what might be considered as bottom-up as well as top-down influences [21].

The interrupt-based account is a close relative of existing notions of phasic NE. For instance, NE has been implicated in the process of alerting [23]. The difference from our account is perhaps the stronger tie in the latter to actual behavioral output. A task with second-order contingencies may help to differentiate the two accounts. There are also close relations with theories [20, 5] that suggest how NE may be an integral part of an optimal decisional strategy. These propose that NE controls the gain in competitive decision-making networks that implement sequential decision-making [22], essentially by reporting on the changes in the statistical structure of the inputs induced by stimulus onset. It is also suggested that a more extreme change in the gain, destabilizing the competitive networks through explosive symmetry breaking, can be used to freeze or lock-in any small difference in the competing activities.

The idea that NE can signal the change in the input statistics occasioned by the (temporally-unpredictable) occurrence of the target is highly appealing. However, the statistics of the input change when either the target *or* the distractor appears, and so the preference for responding to the target at the expense of the distractor is strange. The effect of forcing the decision making network to become unstable, and therefore enforcing a speeded decision is much closer to an interrupt; but then it is not clear why this signal should decrease as the target becomes more common. Further, since in the unstable regime, the statistical optimality of integration is effectively abandoned, the computational appeal of the signal is somewhat weakened. However, this alternative theory does make an important link to sequential statistical analysis [22], raising issues about things like thresholds for deciding target and distractor that should be important foci of future work here too.

Figure 1C shows an additional phenomenon that has arisen in a task when subjects were not even occasionally taxed with difficult discrimination problems. The overall performance fluctuates dramatically (shown by the changing false alarm rate), in a manner that is tightly correlated with fluctuations in tonic NE activity. Periods of high tonic activity are correlated with low phasic activation to the targets (data not shown). Aston-Jones, Cohen and their colleagues [20, 3] have suggested that NE regulates the balance between exploration and exploitation. The high tonic phase is associated with the former, with subjects failing to concentrate on the contingencies that lead to their current rewards in order to search for stimuli or actions that might be associated with better rewards. Increasing the ease of interruptability to either external cues or internal state changes, could certainly lead to apparently exploratory behavior. However, there is little evidence as to how this sort of exploration is being actively determined, since, for instance, the macroscopic fluctuations evident in Figure 1C do not arise in response to any experimental contingency. Given the relationship between phasic and tonic firing, further investigation of these periodic fluctuations and their implications would be desirable.

Finally, in our previous model [24, 25], tonic NE was closely coupled with tonic acetylcholine (ACh), with the latter reporting expected rather than unexpected uncertainty. The account of ACh should transfer somewhat directly into the short-term contingencies within a task – we might expect it to be involved in reporting on aspects of the known variability associated with each state, including each distinct stimulus state as well as the no-stimulus state. As such, this ACh signal might be expected to be relatively more tonic than NE (an effect that is also apparent in our previous work on more tonic interactions between ACh and NE (*eg* Figure 2 of [24]). One attractive target for an account along these lines is the sustained attention task studied by Sarter and colleagues, which involves temporal uncertainty. Performance in this task is exquisitely sensitive to cholinergic manipulation [19], but unaffected by gross noradrenergic manipulation [15]. We may again expect there to be interesting part-opponent and part-synergistic interactions between the neuromodulators.

## Acknowledgements

We are grateful to Gary Aston-Jones, Sebastien Bouret, Jonathan Cohen, Peter Latham, Susan Sara, and Eric Shea-Brown for helpful discussions. Funding was from the Gatsby Charitable Foundation, the EU BIBA project and the ACI Neurosciences Intégratives et Computationnelles of the French Ministry of Research.

## References

[1] Aston-Jones, G, Rajkowski, J, Kubiak, P & Alexinsky, T (1994). Locus coeruleus neurons in monkey are selectively activated by attended cues in a vigilance task. *J. Neurosci.* **14**:4467-4480.

[2] Aston-Jones, G, Rajkowski, J & Kubiak, P (1997). Conditioned responses of monkey locus coeruleus neurons anticipate Acquisition of discriminative behavior in a vigilance task. *Neuroscience* **80**:697-715.

[3] Aston-Jones, G, Rajkowski, J & Cohen, J (2000). Locus coeruleus and regulation of behavioral flexibility and attention. *Prog. Brain Res.* **126**:165-182.

[4] Bouret, S & Sara, SJ (2004). Reward expectation, orientation of attention and locus coeruleus-medial frontal cortex interplay during learning. *Eur. J. Neurosci.* **20**:791-802.

[5] Brown, E, Gao, J, Holmes, P, Bogacz, R, Gilzenrat, M & Cohen, JD (2005). Simple neural networks that optimize decisions. *Int. J. Bif. & Chaos*, in press.

[6] David Johnson, J (2003). Noradrenergic control of cognition: global attenuation and an interrupt function. *Med. Hypoth.* **60**:689-692.

[7] Dayan, P & Yu, AJ (2001). ACh, uncertainty, and cortical inference. *NIPS 2001*.

[8] Desimone, R & Duncan, J (1995). Neural mechanisms of selective visual attention. *Annual Reviews in Neuroscience* **18**:193-222.

[9] Devauges, V & Sara, SJ (1990). Activation of the noradrenergic system facilitates an attentional shift in the rat. *Beh. Brain Res.* **39**:19-28.

[10] Doya, K (2002). Metalearning and neuromodulation. *Neur. Netw.* **15**:495-506.

[11] Foote, SL, Freedman, R & Oliver, AP (1975). Effects of putative neurotransmitters on neuronal activity in monkey auditory cortex. *Brain Res.* **86**:229-242.

[12] Freedman, R, Foote, SL & Bloom, FE (1975) Histochemical characterization of a neocortical projection of the nucleus locus coeruleus in the squirrel monkey. *J. Comp. Neurol.* **164**:209-231.

[13] Jouvet, M (1969). Biogenic amines and the states of sleep. *Science* **163**:32-41.

[14] Marder, E & Thirumalai, V (2002). Cellular, synaptic and network effects of neuromodulation. *Neur. Netw.* **15**:479-493.

[15] McGaughy, J, Sandstrom, M, Ruland, S, Bruno JP & Sarter, M (1997). Lack of effects of lesions of the dorsal noradrenergic bundle on behavioral vigilance. *Beh. Neurosci.* **111**:646-652.

[16] Rajkowski, J, Majczynski, H, Clayton, E & Aston-Jones, G (2004). Activation of monkey locus coeruleus neurons varies with difficulty and performance in a target detection task. *J. Neurophysiol.* **92**:361-371.

[17] Rajkowski, J, Majczynski, H, Clayton, E, Cohen, JD & Aston-Jones, G (2002). Phasic activation of monkey locus coeruleus (LC) neurons with recognition of motivationally relevant stimuli. *Society for Neuroscience, Abstracts* 86.10.

[18] Sara, SJ & Segal, M (1991). Plasticity of sensory responses of locus coeruleus neurons in the behaving rat: implications for cognition. *Prog. Brain Res.* **88**:571-585.

[19] Turchi, J & Sarter, M (2001). Bidirectional modulation of basal forebrain NMDA receptor function differentially affects visual attention but not visual discrimination performance. *Neuroscience* **104**:407-417.

[20] Usher, M, Cohen, JD, Servan-Schreiber, D, Rajkowski, J & Aston-Jones, G (1999). The role of locus coeruleus in the regulation of cognitive performance. *Science* **283**:549-554.

[21] Van Bockstaele, EJ, Chan, J & Pickel, VM (1996). Input from central nucleus of the amygdala efferents to pericoerulear dendrites, some of which contain tyrosine hydroxylase immunoreactivity. *Journal of Neuroscience Research* **45**:289-302.

[22] Wald, A (1947). *Sequential Analysis.* New York, NY: John Wiley & Sons.

[23] Witte, EA & Marrocco, RT (1997). Alteration of brain noradrenergic activity in rhesus monkeys affects the alerting component of covert orienting. *Psychopharmacology* **132**:315-323.

[24] Yu, AJ & Dayan, P (2003). Expected and unexpected uncertainty. ACh and NE in the neocortex. *NIPS 2002*.

[25] Yu, AJ & Dayan, P (2005). Uncertainty, neuromodulation, and attention. *Neuron* **46**, 681-692.
